# Holographic Recurrent Networks

**Tony A. Plate**
Department of Computer Science
University of Toronto
Toronto, M5S 1A4 Canada

## Abstract

Holographic Recurrent Networks (HRNs) are recurrent networks which incorporate associative memory techniques for storing sequential structure. HRNs can be easily and quickly trained using gradient descent techniques to generate sequences of discrete outputs and trajectories through continuous space. The performance of HRNs is found to be superior to that of ordinary recurrent networks on these sequence generation tasks.

## 1  INTRODUCTION

The representation and processing of data with complex structure in neural networks remains a challenge. In a previous paper [Plate, 1991b] I described Holographic Reduced Representations (HRRs) which use circular-convolution associative-memory to embody sequential and recursive structure in fixed-width distributed representations. This paper introduces Holographic Recurrent Networks (HRNs), which are recurrent nets that incorporate these techniques for generating sequences of symbols or trajectories through continuous space. The recurrent component of these networks uses convolution operations rather than the logistic-of-matrix-vector-product traditionally used in simple recurrent networks (SRNs) [Elman, 1991, Cleeremans et al., 1991].

The goals of this work are threefold: (1) to investigate the use of circular-convolution associative memory techniques in networks trained by gradient descent; (2) to see whether adapting representations can improve the capacity of HRRs; and (3) to compare performance of HRNs with SRNs.

## 1.1   RECURRENT NETWORKS & SEQUENTIAL PROCESSING

SRNs have been used successfully to process sequential input and induce finite state grammars [Elman, 1991, Cleeremans *et al.*, 1991]. However, training times were extremely long, even for very simple grammars. This appeared to be due to the difficulty of finding a recurrent operation that preserved sufficient context [Maskara and Noetzel, 1992]. In the work reported in this paper the task is reversed to be one of generating sequential output. Furthermore, in order to focus on the context retention aspect, no grammar induction is required.

## 1.2   CIRCULAR CONVOLUTION

Circular convolution is an associative memory operator. The role of convolution in holographic memories is analogous to the role of the outer product operation in matrix style associative memories (e.g., Hopfield nets). Circular convolution can be viewed as a vector multiplication operator which maps pairs of vectors to a vector (just as matrix multiplication maps pairs of matrices to a matrix). It is defined as $\mathbf{z} = \mathbf{x} \circledast \mathbf{y} : z_j = \sum_{k=0}^{n-1} y_k x_{j-k}$, where $\circledast$ denotes circular convolution, $\mathbf{x}$, $\mathbf{y}$, and $\mathbf{z}$ are vectors of dimension $n$, $x_i$ etc. are their elements, and subscripts are modulo-n (so that $x_{-2} \equiv x_{n-2}$). Circular convolution can be computed in $O(n \log n)$ using Fast Fourier Transforms (FFTs). Algebraically, convolution behaves like scalar multiplication: it is commutative, associative, and distributes over addition. The identity vector for convolution ($\mathbf{I}$) is the "impulse" vector: its zero'th element is 1 and all other elements are zero. Most vectors have an inverse under convolution, i.e., for most vectors $\mathbf{x}$ there exists a unique vector $\mathbf{y}$ ($= \mathbf{x}^{-1}$) such that $\mathbf{x} \circledast \mathbf{y} = \mathbf{I}$. For vectors with identically and independently distributed zero mean elements and an expected Euclidean length of 1 there is a numerically stable and simply derived approximate inverse. The approximate inverse of $\mathbf{x}$ is denoted by $\mathbf{x}^*$ and is defined by the relation $x_j^* = x_{n-j}$.

Vector pairs can be associated by circular convolution. Multiple associations can be summed. The result can be decoded by convolving with the exact inverse or approximate inverse, though the latter generally gives more stable results.

Holographic Reduced Representations [Plate, 1991a, Plate, 1991b] use circular convolution for associating elements of a structure in a way that can embody hierarchical structure. The key property of circular convolution that makes it useful for representing hierarchical structure is that the circular convolution of two vectors is another vector of the same dimension, which can be used in further associations.

Among associative memories, holographic memories have been regarded as inferior because they produce very noisy results and have poor error correcting properties. However, when used in Holographic Reduced Representations the noisy results can be cleaned up with conventional error correcting associative memories. This gives the best of both worlds – the ability to represent sequential and recursive structure and clean output vectors.

## 2   TRAJECTORY-ASSOCIATION

A simple method for storing sequences using circular convolution is to associate elements of the sequence with points along a predetermined trajectory. This is akin

to the memory aid called the method of loci which instructs us to remember a list of items by associating each term with a distinctive location along a familiar path.

## 2.1  STORING SEQUENCES BY TRAJECTORY-ASSOCIATION

Elements of the sequence and loci (points) on the trajectory are all represented by n-dimensional vectors. The loci are derived from a single vector $\mathbf{k}$ – they are its successive convolutive powers: $\mathbf{k}^0$, $\mathbf{k}^1$, $\mathbf{k}^2$, etc. The convolutive power is defined in the obvious way: $\mathbf{k}^0$ is the identity vector and $\mathbf{k}^{i+1} = \mathbf{k}^i \circledast \mathbf{k}$.

The vector $\mathbf{k}$ must be chosen so that it does not blow up or disappear when raised to high powers, i.e., so that $\|\mathbf{k}^p\| = 1 \ \forall \ p$. The class of vectors which satisfy this constraint is easily identified in the frequency domain (the range of the discrete Fourier transform). They are the vectors for which the magnitude of the power of each frequency component is equal to one. This class of vectors is identical to the class for which the approximate inverse is equal to the exact inverse.

Thus, the trajectory-association representation for the sequence "abc" is

$$s_{abc} = \mathbf{a} + \mathbf{b} \circledast \mathbf{k} + \mathbf{c} \circledast \mathbf{k}^2.$$

## 2.2  DECODING TRAJECTORY-ASSOCIATED SEQUENCES

Trajectory-associated sequences can be decoded by repeatedly convolving with the inverse of the vector that generated the encoding loci. The results of decoding summed convolution products are very noisy. Consequently, to decode trajectory associated sequences, we must have all the possible sequence elements stored in an error correcting associative memory. I call this memory the "clean up" memory.

For example, to retrieve the third element of the sequence $s_{abc}$ we convolve twice with $\mathbf{k}^{-1}$, which expands to $\mathbf{a} \circledast \mathbf{k}^{-2} + \mathbf{b} \circledast \mathbf{k}^{-1} + \mathbf{c}$. The two terms involving powers of $\mathbf{k}$ are unlikely to be correlated with anything in the clean up memory. The most similar item in clean up memory will probably be $\mathbf{c}$. The clean up memory should recognize this and output the clean version of $\mathbf{c}$.

## 2.3  CAPACITY OF TRAJECTORY-ASSOCIATION

In [Plate, 1991a] the capacity of circular-convolution based associative memory was calculated. It was assumed that the elements of all vectors (dimension $n$) were chosen randomly from a gaussian distribution with mean zero and variance $1/n$ (giving an expected Euclidean length of 1.0). Quite high dimensional vectors were required to ensure a low probability of error in decoding. For example, with 512 element vectors and 1000 items in the clean up memory, 5 pairs can be stored with a 1% chance of an error in decoding. The scaling is nearly linear in $n$: with 1024 element vectors 10 pairs can be stored with about a 1% chance of error. This works out to a information capacity of about 0.1 bits per element. The elements are real numbers, but high precision is not required.

These capacity calculations are roughly applicable to the trajectory-association method. They slightly underestimate its capacity because the restriction that the encoding loci have unity power in all frequencies results in lower decoding noise. Nonetheless this figure provides a useful benchmark against which to compare the capacity of HRNs which adapt vectors using gradient descent.

# 3   TRAJECTORY ASSOCIATION & RECURRENT NETS

HRNs incorporate the trajectory-association scheme in recurrent networks. HRNs are very similar to SRNs, such as those used by [Elman, 1991] and [Cleeremans *et al.*, 1991]. However, the task used in this paper is different: the generation of target sequences at the output units, with inputs that do not vary in time.

In order to understand the relationship between HRNs and SRNs both were tested on the sequence generation task. Several different unit activation functions were tried for the SRN: symmetric (tanh) and non-symmetric sigmoid ($1/(1 + e^{-x})$) for the hidden units, and softmax and normalized RBF for the output units. The best combination was symmetric sigmoid with softmax outputs.

## 3.1   ARCHITECTURE

The HRN and the SRN used in the experiments described here are shown in Figure 1. In the HRN the key layer $\mathbf{y}$ contains the generator for the inverse loci (corresponding to $\mathbf{k}^{-1}$ in Section 2). The hidden to output nodes implement the clean-up memory: the output representation is local and the weights on the links to an output unit form the vector that represents the symbol corresponding to that unit. The softmax function serves to give maximum activation to the output unit whose weights are most similar to the activation at the hidden layer.

The input representation is also local, and input activations do not change during the generation of one sequence. Thus the weights from a single input unit determine the activations at the code layer. Nets are reset at the beginning of each sequence.

The HRN computes the following functions. Time superscripts are omitted where all are the same. See Figure 1 for symbols. The parameter $g$ is an adaptable input gain shared by all output units.

| | | | |
|---|---|---|---|
| Code units: | | $c_j = \sum_k i_k w_{jk}^c$ | |
| Hidden units: | (first time step) | $h_j = c_j$ | |
| | (subsequent steps) | $h_j = \sum_k p_k y_{j-k}$ | $(\mathbf{h} = \mathbf{p} \circledast \mathbf{y})$ |
| Context units: | | $p_j^{t+1} = p_j^t$ | |
| Output units: | (total input) | $x_j = g \sum_k h_k w_{jk}^o$ | |
| | (output) | $o_j = \dfrac{e^{x_j}}{\sum_k e^{x_k}}$ | (softmax) |

In the SRN the only difference is in the recurrence operation, i.e., the computation of the activations of the hidden units which is, where $b_j$ is a bias:

$$h_j = \tanh(c_j + \sum_k w_{jk}^r p_k + b_j).$$

The objective function of the network is the asymmetric divergence between the activations of the output units ($o_j^{st}$) and the targets ($t_j^{st}$) summed over cases $s$ and timesteps $t$, plus two weight penalty terms ($n$ is the number of hidden units):

$$E = - \left( \sum_{stj} t_j^{st} \log \frac{o_j^{st}}{t_j^{st}} \right) + \frac{0.0001}{n} \left( \sum_{jk} w_{jk}^r + \sum_{jk} w_{jk}^c \right) + \sum_j \left( 1 - \sum_k {w_{jk}^o}^2 \right)^2$$

The first weight penalty term is a standard weight cost designed to penalize large

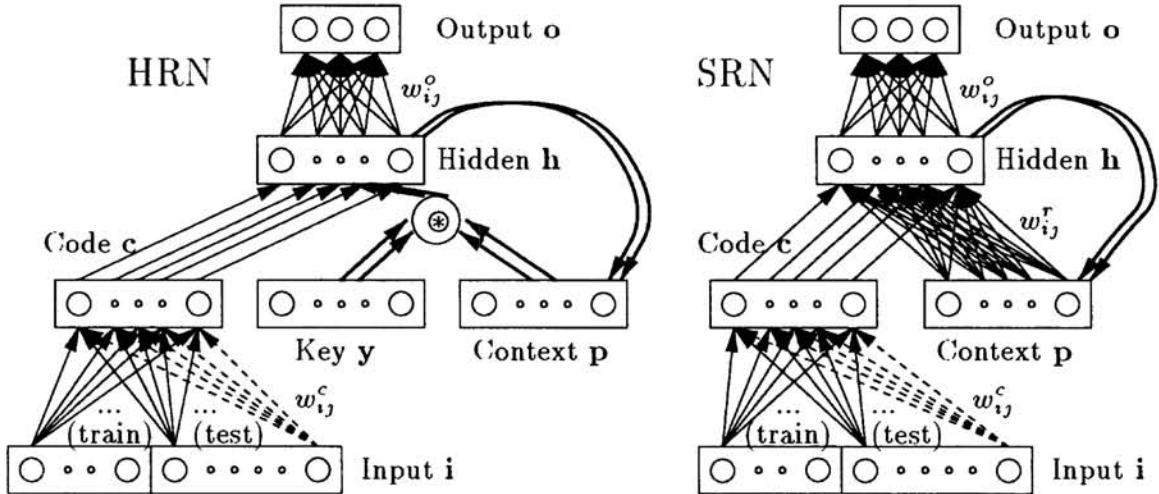

Figure 1: Holographic Recurrent Network (HRN) and Simple Recurrent Network (SRN). The backwards curved arrows denote a copy of activations to the next time step. In the HRN the code layer is active only at the first time step and the context layer is active only after the first time step. The hidden, code, context, and key layers all have the same number of units. Some input units are used only during training, others only during testing.

weights. The second weight penalty term was designed to force the Euclidean length of the weight vector on each output unit to be one. This penalty term helped the HRN considerably but did not noticeably improve the performance of the SRN.

The partial derivatives for the activations were computed by the unfolding in time method [Rumelhart *et al.*, 1986]. The partial derivatives for the activations of the context units in the HRN are:

$$\frac{\partial E}{\partial p_j} = \sum_k \frac{\partial E}{\partial h_j} y_{k-j} \qquad (\equiv \nabla_{\mathbf{p}} E = \nabla_{\mathbf{h}} \circledast \mathbf{y}^*)$$

When there are a large number of hidden units it is more efficient to compute this derivative via FFTs as the convolution expression on the right.

On all sequences the net was cycled for as many time steps as required to produce the target sequence. The outputs did not indicate when the net had reached the end of the sequence, however, other experiments have shown that it is a simple matter to add an output to indicate this.

## 3.2   TRAINING AND GENERATIVE CAPACITY RESULTS

One of the motivations for this work was to find recurrent networks with high generative capacity, i.e., networks which after training on just a few sequences could generate many other sequences without further modification of recurrent or output weights. The only thing in the network that changes to produce a different sequence is the activation on the codes units. To have high generative capacity the function of the output weights and recurrent weights (if they exist) must generalize to the production of novel sequences. At each step the recurrent operation must update and retain information about the current position in the sequence. It was

expected that this would be a difficult task for SRNs, given the reported difficulties with getting SRNs to retain context, and Simard and LeCun's [1992] report of being unable to train a type of recurrent network to generate more than one trajectory through continuous space. However, it turned out that HRNs, and to a lesser extent SRNs, could be easily trained to perform the sequence generation task well.

The generative capacity of HRNs and SRNs was tested using randomly chosen sequences over 3 symbols (a, b, and c). The training data was (in all but one case) 12 sequences of length 4, e.g., "abac", and "bacb". Networks were trained on this data using the conjugate gradient method until all sequences were correctly generated. A symbol was judged to be correct when the activation of the correct output unit exceeded 0.5 and exceeded twice any other output unit activation.

After the network had been trained, all the weights and parameters were frozen, except for the weights on the input to code links. Then the network was trained on a test set of novel sequences of lengths 3 to 16 (32 sequences of each length). This training could be done one sequence at a time since the generation of each sequence involved an exclusive set of modifiable weights, as only one input unit was active for any sequence. The search for code weights for the test sequences was a conjugate gradient search limited to 100 iterations.

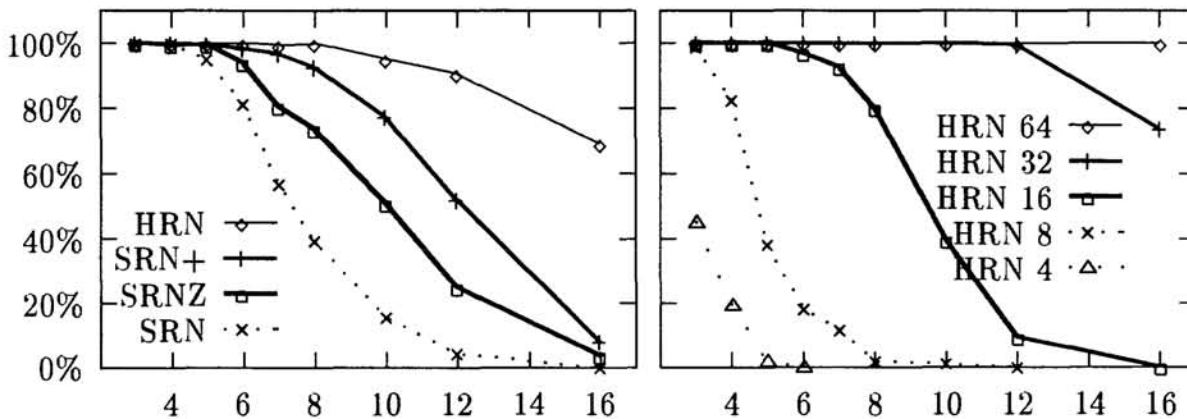

Figure 2: Percentage of novel sequences that can be generated versus length.

The graph on the left in Figure 2 shows how the performance varies with sequence length for various networks with 16 hidden units. The points on this graph are the average of 5 runs; each run began with a randomization of all weights. The worst performance was produced by the SRN. The HRN gave the best performance: it was able to produce around 90% of all sequences up to length 12. Interestingly, a SRN (SRNZ in Figure 2) with frozen random recurrent weights from a suitable distribution performed significantly better than the unconstrained SRN.

To some extent, the poor performance of the SRN was due to overtraining. This was verified by training a SRN on 48 sequences of length 8 (8 times as much data). The performance improved greatly (SRN+ in Figure 2), but was still not as good that of the HRN trained on the lesser amount of data. This suggests that the extra parameters provided by the recurrent links in the SRN serve little useful purpose: the net does well with fixed random values for those parameters and a HRN does better without modifying any parameters in this operation. It appears that all that

is required in the recurrent operation is some stable random map.

The scaling performance of the HRN with respect to the number of hidden units is good. The graph on the right in Figure 2 shows the performance of HRNs with 8 output units and varying numbers of hidden units (averages of 5 runs). As the number of hidden units increases from 4 to 64 the generative capacity increases steadily. The scaling of sequence length with number of outputs (not shown) is also good: it is over 1 bit per hidden unit. This compares very will with the 0.1 bit per element achieved by random vector circular-convolution (Section 2.3).

The training times for both the HRNs and the SRNs were very short. Both required around 30 passes through the training data to train the output and recurrent weights. Finding a code for test sequence of length 8 took the HRN an average of 14 passes. The SRN took an average of 57 passes (44 with frozen weights). The SRN trained on more data took much longer for the initial training (average 281 passes) but the code search was shorter (average 31 passes).

## 4    TRAJECTORIES IN CONTINUOUS SPACE

HRNs can also be used to generate trajectories through continuous space. Only two modifications need be made: (a) change the function on the output units to sigmoid and add biases, and (b) use a fractional power for the key vector. A fractional power vector $\mathbf{f}$ can be generated by taking a random unity-power vector $\mathbf{k}$ and multiplying the phase angle of each frequency component by some fraction $\alpha$, i.e., $\mathbf{f} = \mathbf{k}^{\alpha}$. The result is that $\mathbf{f}^i$ is similar to $\mathbf{f}^j$ when the difference between $i$ and $j$ is less than $1/\alpha$, and the similarity is greater for closer $i$ and $j$. The output at the hidden layer will be similar at successive time steps. If desired, the speed at which the trajectory is traversed can be altered by changing $\alpha$.

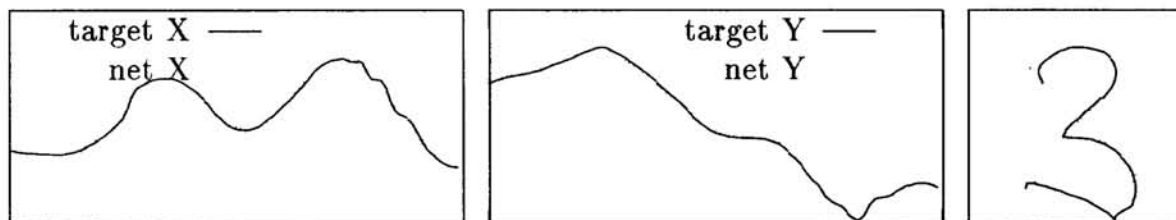

Figure 3: Targets and outputs of a HRN trained to generate trajectories through continuous space. X and Y are plotted against time.

A trajectory generating HRN with 16 hidden units and a key vector $\mathbf{k}^{0.06}$ was trained to produce pen trajectories (100 steps) for 20 instances of handwritten digits (two of each). This is the same task that Simard and Le Cun [1992] used. The target trajectories and the output of the network for one instance are shown in Figure 3.

## 5    DISCUSSION

One issue in processing sequential data with neural networks is how to present the inputs to the network. One approach has been to use a fixed window on the sequence, e.g., as in NETtalk [Sejnowski and Rosenberg, 1986]. A disadvantage of this is any fixed size of window may not be large enough in some situations. Another approach is to use a recurrent net to retain information about previous

inputs. A disadvantage of this is the difficulty that recurrent nets have in retaining information over many time steps. Generative networks offer another approach: use the codes that generate a sequence as input rather than the raw sequence. This would allow a fixed size network to take sequences of variable length as inputs (as long as they were finite), without having to use multiple input blocks or windows.

The main attraction of circular convolution as an associative memory operator is its affordance of the representation of hierarchical structure. A hierarchical HRN, which takes advantage of this to represent sequences in chunks, has been built. However, it remains to be seen if it can be trained by gradient descent.

## 6    CONCLUSION

The circular convolution operation can be effectively incorporated into recurrent nets and the resulting nets (HRNs) can be easily trained using gradient descent to generate sequences and trajectories. HRNs appear to be more suited to this task than SRNs, though SRNs did surprisingly well. The relatively high generative capacity of HRNs shows that the capacity of circular convolution associative memory [Plate, 1991a] can be greatly improved by adapting representations of vectors.

## References

[Cleeremans *et al.*, 1991] A. Cleeremans, D. Servan-Schreiber, and J. L. McClelland. Graded state machines: The representation of temporal contingencies in simple recurrent networks. *Machine Learning*, 7(2/3):161–194, 1991.

[Elman, 1991] J. Elman. Distributed representations, simple recurrent networks and grammatical structure. *Machine Learning*, 7(2/3):195–226, 1991.

[Maskara and Noetzel, 1992] Arun Maskara and Andrew Noetzel. Forcing simple recurrent neural networks to encode context. In *Proceedings of the 1992 Long Island Conference on Artificial Intelligence and Computer Graphics*, 1992.

[Plate, 1991a] T. A. Plate. Holographic Reduced Representations. Technical Report CRG-TR-91-1, Department of Computer Science, University of Toronto, 1991.

[Plate, 1991b] T. A. Plate. Holographic Reduced Representations: Convolution algebra for compositional distributed representations. In *Proceedings of the 12th International Joint Conference on Artificial Intelligence*, pages 30–35, Sydney, Australia, 1991.

[Rumelhart *et al.*, 1986] D. E. Rumelhart, G. E. Hinton, and Williams R. J. Learning internal representations by error propagation. In *Parallel distributed processing: Explorations in the microstructure of cognition*, volume 1, chapter 8, pages 318–362. Bradford Books, Cambridge, MA, 1986.

[Sejnowski and Rosenberg, 1986] T. J. Sejnowski and C. R. Rosenberg. *NETtalk: A parallel network that learns to read aloud*. Technical report 86-01, Department of Electrical Engineering and Computer Science, Johns Hopkins University, Baltimore, MD., 1986.

[Simard and LeCun, 1992] P. Simard and Y. LeCun. Reverse TDNN: an architecture for trajectory generation. In J. M. Moody, S. J. Hanson, and R. P. Lippman, editors, *Advances in Neural Information Processing Systems 4 (NIPS*91)*, Denver, CO, 1992. Morgan Kaufman.
